# A Connectionist Model for Constructive Modal Reasoning

**Artur S. d'Avila Garcez**
Department of Computing, City University London
London EC1V 0HB, UK
aag@soi.city.ac.uk

**Luís C. Lamb**
Institute of Informatics, Federal University of Rio Grande do Sul
Porto Alegre RS, 91501-970, Brazil
LuisLamb@acm.org

**Dov M. Gabbay**
Department of Computer Science, King's College London
Strand, London, WC2R 2LS, UK
dg@dcs.kcl.ac.uk

## Abstract

We present a new connectionist model for constructive, intuitionistic modal reasoning. We use ensembles of neural networks to represent intuitionistic modal theories, and show that for each intuitionistic modal program there exists a corresponding neural network ensemble that computes the program. This provides a massively parallel model for intuitionistic modal reasoning, and sets the scene for integrated reasoning, knowledge representation, and learning of intuitionistic theories in neural networks, since the networks in the ensemble can be trained by examples using standard neural learning algorithms.

## 1   Introduction

Automated reasoning and learning theory have been the subject of intensive investigation since the early developments in computer science [14]. However, while (machine) learning has focused mainly on quantitative and connectionist approaches [16], the reasoning component of intelligent systems has been developed mainly by formalisms of classical and non-classical logics [7, 9]. More recently, the recognition of the need for systems that integrate reasoning and learning into the same foundation, and the evolution of the fields of cognitive and neural computation, has led to a number of proposals that attempt to integrate reasoning and learning [1, 3, 12, 13, 15].

We claim that an effective integration of reasoning and learning can be obtained by neural-symbolic learning systems [3, 4]. Such systems concern the application of problem-specific symbolic knowledge within the neurocomputing paradigm. By integrating logic and neural

networks, they may provide ($i$) a sound logical characterisation of a connectionist system, ($ii$) a connectionist (parallel) implementation of a logic, or ($iii$) a hybrid learning system bringing together advantages from connectionism and symbolic reasoning.

Intuitionistic logical systems have been advocated by many as providing adequate logical foundations for computation (see [2] for a survey). We argue, therefore, that intuitionism could also play an important part in neural computation. In this paper, we follow the research path outlined in [4, 5], and develop a computational model for integrated reasoning, representation, and learning of intuitionistic modal knowledge. We concentrate on reasoning and knowledge representation issues, which set the scene for connectionist intuitionistic learning, since effective knowledge representation should precede learning [15]. Still, we base the representation on standard, simple neural network architectures, aiming at future work on experimental learning within the model proposed here.

A key contribution of this paper is the proposal to shift the notion of logical implication (and negation) in neural networks from the standard notion of implication as a partial function from input to output (and of negation as failure to activate a neuron), to an intuitionistic notion which we will see can be implemented in neural networks if we make use of network ensembles. We claim that the intuitionistic interpretation introduced here will make sense for a number of problems in neural computation in the same way that intuitionistic logic is more appropriate than classical logic in a number of computational settings. We will start by illustrating the proposed computational model in an appropriate constructive reasoning, distributed knowledge representation scenario, namely, the *wise men puzzle* [7]. Then, we will show how ensembles of *Connectionist Inductive Learning and Logic Programming* (C-ILP) networks [3] can compute intuitionistic modal knowledge. The networks are set up by an *Intuitionistic Modal Algorithm* introduced in this paper. A proof that the algorithm produces a neural network ensemble that computes a semantics of its associated intuitionistic modal theory is then given. Furthermore, the networks in the ensemble are kept simple and in a modular structure, and may be trained from examples with the use of standard learning algorithms such as *backpropagation* [11].

In Section 2, we present the basic concepts of intuitionistic reasoning used in the paper. In Section 3, we motivate the proposed model using the wise men puzzle. In Section 4, we introduce the *Intuitionistic Modal Algorithm*, which translates intuitionistic modal theories into neural network ensembles, and prove that the ensemble computes a semantics of the theory. Section 5 concludes the paper and discusses directions for future work.

## 2 Background

In this section, we present some basic concepts of artificial neural networks and intuitionistic programs used throughout the paper. We concentrate on ensembles of single hidden layer feedforward networks, and on recurrent networks typically with feedback only from the output to the input layer. Feedback is used with the sole purpose of denoting that the output of a neuron should serve as the input of another neuron when we run the network, i.e. the weight of any feedback connection is fixed at 1. We use *bipolar* semi-linear activation functions $h(x) = \frac{2}{1+e^{-\beta x}} - 1$ with inputs in $\{-1, 1\}$. Throughout, we will use 1 to denote truth-value $true$, and $-1$ to denote truth-value $false$.

Intuitionistic logic was originally developed by Brouwer, and later by Heyting and Kolmogorov [2]. In intuitionistic logics, a statement that there exists a proof of a proposition $x$ is only made if there is a constructive method of the proof of $x$. One of the consequences of Brouwer's ideas is the rejection of the law of the excluded middle, namely $\alpha \lor \neg \alpha$, since one cannot always state that there is a proof of $\alpha$ or of its negation, as accepted in classical logic and in (classical) mathematics. The development of these ideas and applications in mathematics has led to developments in *constructive* mathematics and has influenced

several lines of research on logic and computing science [2].

An intuitionistic modal language $\mathcal{L}$ includes propositional letters (atoms) $p, q, r...$, the connectives $\neg, \wedge$, an intuitionistic implication $\Rightarrow$, the *necessity* ($\Box$) and *possibility* ($\Diamond$) modal operators, where an atom will be necessarily true in a possible world if it is true in every world that is related to this possible world, while it will be possibly true if it is true in some world related to this world. Formally, we interpret the language as follows, where formulas are denoted by $\alpha, \beta, \gamma...$

**Definition 1** *(*Kripke Models for Intuitionistic Modal Logic*) Let $\mathcal{L}$ be an intuitionistic language. A* model *for $\mathcal{L}$ is a tuple $\mathcal{M} = \langle \Omega, \mathcal{R}, v \rangle$ where $\Omega$ is a set of worlds, $v$ is a mapping that assigns to each $\omega \in \Omega$ a subset of the atoms of $\mathcal{L}$, and $\mathcal{R}$ is a reflexive, transitive, binary relation over $\Omega$, such that: (a) $(\mathcal{M}, \omega) \models p$ iff $p \in v(\omega)$ (for atom $p$); (b) $(\mathcal{M}, \omega) \models \neg\alpha$ iff for all $\omega'$ such that $\mathcal{R}(\omega, \omega')$, $(\mathcal{M}, \omega') \not\models \alpha$; (c) $(\mathcal{M}, \omega) \models \alpha \wedge \beta$ iff $(\mathcal{M}, \omega) \models \alpha$ and $(\mathcal{M}, \omega) \models \beta$; (d) $(\mathcal{M}, \omega) \models \alpha \Rightarrow \beta$ iff for all $\omega'$ with $\mathcal{R}(\omega, \omega')$ we have $(\mathcal{M}, \omega') \models \beta$ whenever we have $(\mathcal{M}, \omega') \models \alpha$; (e) $(\mathcal{M}, \omega) \models \Box\alpha$ iff for all $\omega' \in \Omega$ if $\mathcal{R}(\omega, \omega')$ then $(\mathcal{M}, \omega') \models \alpha$; (f) $(\mathcal{M}, \omega) \models \Diamond\alpha$ iff there exists $\omega' \in \Omega$ such that $\mathcal{R}(\omega, \omega')$ and $(\mathcal{M}, \omega') \models \alpha$.*

We now define *labelled intuitionistic programs* as sets of intuitionistic rules, where each rule is labelled by the world at which it holds, similarly to Gabbay's Labelled Deductive Systems [8].

**Definition 2** *(*Labelled Intuitionistic Program*) A Labelled Intuitionistic Program is a finite set of rules $C$ of the form $\omega_i : A_1, ..., A_n \Rightarrow A_0$ (where "," abbreviates "$\wedge$", as usual), and a finite set of relations $\mathcal{R}$ between worlds $\omega_i$ ($1 \leq i \leq m$) in $C$, where $A_k$ ($0 \leq k \leq n$) are atoms and $\omega_i$ is a label representing a world in which the associated rule holds.*

To deal with intuitionistic negation, we adopt the approach of [10], as follows. We rename any negative literal $\neg A$ as an atom $A'$ not present originally in the language. This form of renaming allows our definition of labelled intuitionistic programs above to consider atoms only. For example, given $A_1, ..., A'_k, ..., A_n \Rightarrow A_0$, where $A'_k$ is a renaming of $\neg A_k$, an interpretation that assigns true to $A'_k$ represents that $\neg A_k$ is true; it does not represent that $A_k$ is false. Following Definition 1 (intuitionistic negation), $A'$ will be true in a world $\omega_i$ if and only if $A$ does not hold in every world $\omega_j$ such that $\mathcal{R}(\omega_i, \omega_j)$.
Finally, we extend labelled intuitionistic programs to include modalities.

**Definition 3** *(*Labelled Intuitionistic Modal Program*) A modal atom is of the form $MA$ where $M \in \{\Box, \Diamond\}$ and $A$ is an atom. A Labelled Intuitionistic Modal Program is a finite set of rules $C$ of the form $\omega_i : MA_1, ..., MA_n \Rightarrow MA_0$, where $MA_k$ ($0 \leq k \leq n$) are modal atoms and $\omega_i$ is a label representing a world in which the associated rule holds, and a finite set of (accessibility) relations $\mathcal{R}$ between worlds $\omega_i$ ($1 \leq i \leq m$) in $C$.*

## 3 Motivating Scenario

In this section, we consider an archetypal testbed for distributed knowledge representation, namely, the *wise men puzzle* [7], and model it intuitionistically in a neural network ensemble. Our aim is to illustrate the combination of neural networks and intuitionistic modal reasoning. The formalisation of our computational model will be given in Section 4.

*A certain king wishes to test his three wise men. He arranges them in a circle so that they can see and hear each other. They are all perceptive, truthful and intelligent, and this is common knowledge in the group. It is also common knowledge among them that there are three red hats and two white hats, and five hats in total. The king places a hat on the head*

*of each wise man in a way that they are not able to see the colour of their own hats, and then asks each one whether they know the colour of the hats on their heads.*

The puzzle illustrates a situation in which intuitionistic implication and intuitionistic negation occur. Knowledge evolves in time, with the current knowledge persisting in time. For example, at the first round it is known that there are at most two white hats on the wise men's heads. Then, if the wise men get to a second round, it becomes known that there is at most one white hat on their heads.[1] This new knowledge subsumes the previous knowledge, which in turn persists. This means that if $A \Rightarrow B$ is true at a world $t_1$ then $A \Rightarrow B$ will be true at a world $t_2$ that is related to $t_1$ (intuitionistic implication). Now, in any situation in which a wise man knows that his hat is red, this knowledge - constructed with the use of sound reasoning processes - cannot be refuted. In other words, in this puzzle, if $\neg A$ is true at world $t_1$ then $A$ cannot be true at a world $t_2$ that is related to $t_1$ (intuitionistic negation).

We model the wise men puzzle by constructing the relative knowledge of each wise man along time points. This allows us to explicitly represent the relativistic notion of knowledge, which is a principle of intuitionistic reasoning. For simplicity, we refer to wise man 1 (respectively, 2 and 3) as agent 1 (respectively, 2 and 3). The resulting model is a two-dimensional network ensemble (agents $\times$ time), containing three networks in each dimension. In addition to $p_i$ - denoting the fact that wise man $i$ wears a red hat - to model each agent's individual knowledge, we need to use a modality $K_j$, $j \in \{1, 2, 3\}$, which represents the relative notion of knowledge at each time point $t_1$, $t_2$, $t_3$. Thus, $K_j p_i$ denotes the fact that agent $j$ knows that agent $i$ wears a red hat. The $K$ modality above corresponds to the $\square$ modality in intuitionistic modal reasoning, as customary in the logics of knowledge [7], and as exemplified below.

First, we model the fact that each agent knows the colour of the others' hats. For example, if wise man 3 wears a red hat (neuron $p_3$ is active) then wise man 1 knows that wise man 3 wears a red hat (neuron $K p_3$ is active for wise man 1). We then need to model the reasoning process of each wise man. In this example, let us consider the case in which neurons $p_1$ and $p_3$ are active. For agent 1, we have the rule $t_1 : K_1 \neg p_2 \wedge K_1 \neg p_3 \Rightarrow K_1 p_1$, which states that agent 1 can deduce that he is wearing a red hat if he knows that the other agents are both wearing white hats. Analogous rules exist for agents 2 and 3. As before, the implication is intuitionistic, so that it persists at $t_2$ and $t_3$ as depicted in Figure 1 for wise man 1 (represented via hidden neuron $h_1$ in each network). In addition, according to the philosophy of intuitionistic negation, we may only conclude that agent 1 knows $\neg p_2$, if in every world envisaged by agent 1, $p_2$ is not derived. This is illustrated with the use of dotted lines in Figure 1, in which, e.g., if neuron $K p_2$ is not active at $t_3$ then neuron $K \neg p_2$ will be active at $t_2$. As a result, the network ensemble will never derive $p_2$ (as one should expect), and thus it will derive $K_1 \neg p_2$ and $K_3 \neg p_2$.[2]

## 4  Connectionist Intuitionistic Modal Reasoning

The wise men puzzle example of Section 3 shows that simple, single-hidden layer neural networks can be combined in a modular structure where each network represents a possible world in the Kripke structure of Definition 1. The way that the networks should then be inter-connected can be defined by following a semantics for $\Rightarrow$ and $\neg$, and for $\square$ and $\Diamond$ from intuitionistic logic. In this section, we see how exactly we construct a network ensemble

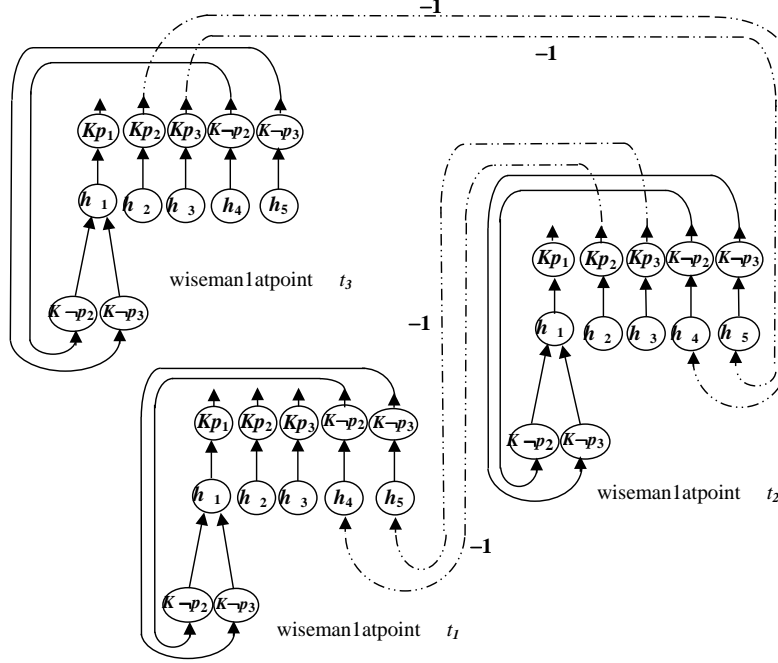

Figure 1: Wise men puzzle: Intuitionistic negation and implication.

given an intuitionistic modal program. We introduce a translation algorithm, which takes the program as input and produces the ensemble as output by setting the initial architecture, set of weights, and thresholds of the networks according to a Kripke semantics for the program. We then prove that the translation is correct, and thus that the network ensemble can be used to compute the logical consequences of the program in parallel.

Before we present the algorithm, let us illustrate informally how $\Rightarrow$, $\neg$, $\Box$, and $\Diamond$ are represented in the ensemble. We follow the key idea behind *Connectionist Modal Logics* (CML) to represent Kripke models in neural networks [6]. Each possible world is represented by a single hidden layer neural network. In each network, input and output neurons represent atoms or modal atoms of the form $A$, $\neg A$, $\Box A$, or $\Diamond A$, while each hidden neuron encodes a rule. For example, in Figure 1, hidden neuron $h_1$ encodes a rule of the form $A \wedge B \Rightarrow C$. Thresholds and weights must be such that the hidden layer computes a logical *and* of the input layer, while the output layer computes a logical *or* of the hidden layer.[3] Furthermore, in each network, each output neuron is connected to its corresponding input neuron with a weight fixed at $1.0$ (as depicted in Figure 1 for $K\neg p_2$ and $K\neg p_3$), so that chains of the form $A \Rightarrow B$ and $B \Rightarrow C$ can be represented and computed. This basically characterises C-ILP networks [3]. Now, in CML, we allow for an ensemble of C-ILP networks, each network representing knowledge in a (learnable) possible world. In addition, we allow for a number of fixed feedforward and feedback connections to occur among different networks in the ensemble, as shown in Figure 1. These are defined as follows: in the case of $\Box$, if neuron $\Box A$ is activated (*true*) in network (world) $\omega_i$ then $A$ must be activated in every network $\omega_j$ that is related to $\omega_i$ (this is analogous to the situation in which we activate $K_1 p_3$ and $K_2 p_3$ whenever $p_3$ is active). Dually, if $A$ is active in every $\omega_j$ then $\Box A$ must be activated

in $\omega_i$ (this is done with the use of feedback connections and a hidden neuron that computes a logical *and*, as detailed in the algorithm below). In the case of $\Diamond$, if $\Diamond A$ is activated in network $\omega_i$ then $A$ must be activated in at least one network $\omega_j$ that is related to $\omega_i$ (we do this by choosing an arbitrary $\omega_j$ to make $A$ active). Dually, if $A$ is activated in any $\omega_j$ that is related to $\omega_i$ then $\Diamond A$ must be activated in $\omega_i$ (this is done with the use of a hidden neuron that computes a logical *or*, also as detailed in the algorithm below). Now, in the case of $\Rightarrow$, according to the semantics of intuitionistic implication, $\omega_i : A \Rightarrow B$ and $\mathcal{R}(\omega_i, \omega_j)$ imply $\omega_j : A \Rightarrow B$. We implement this by copying the neural representation of $A \Rightarrow B$ from $\omega_i$ to $\omega_j$, as done via $h_1$ in Figure 1. Finally, in the case of $\neg$, we need to make sure that $\neg A$ is activated in $\omega_i$ if, for every $\omega_j$ such that $\mathcal{R}(\omega_i, \omega_j)$, $A$ is not active in $\omega_j$. This is implemented with the use of negative weights (to account for the fact that the non-activation of a neuron needs to activate another neuron), as depicted in Figure 1 (dashed arrows), and detailed in the algorithm below.

We are now in a position to introduce the *Intuitionistic Modal Algorithm*. Let $\mathcal{P} = \{\mathcal{P}_1, ..., \mathcal{P}_n\}$ be a labelled intuitionistic modal program with rules of the form $\omega_i : MA_1, ..., MA_k \to MA_0$, where each $A_j$ ($0 \leq j \leq k$) is an atom and $M \in \{\Box, \Diamond\}$, $1 \leq i \leq n$. Let $\mathcal{N} = \{\mathcal{N}_1, ..., \mathcal{N}_n\}$ be a neural network ensemble with each network $\mathcal{N}_i$ corresponding to program $\mathcal{P}_i$. Let $q$ denote the number of rules occurring in $\mathcal{P}$. Consider that the atoms of $\mathcal{P}_i$ are numbered from 1 to $\eta_i$ such that the input and output layers of $\mathcal{N}_i$ are vectors of length $\eta_i$, where the j-th neuron represents the j-th atom of $\mathcal{P}_i$. In addition, let $A_{min}$ denote the minimum activation for a neuron to be considered *active* (or *true*), $A_{min} \in (0, 1)$; for each rule $r_l$ in each program $\mathcal{P}_i$, let $k_l$ denote the number of atoms in the body of rule $r_l$, and let $\mu_l$ denote the number of rules in $\mathcal{P}_i$ with the same consequent as $r_l$ (including $r_l$). Let $MAX_{r_l}(k_l, \mu_l)$ denote the greater of $k_l$ and $\mu_l$ for rule $r_l$, and let $MAX_{\mathcal{P}}(k_1, ..., k_q, \mu_1, ..., \mu_q)$ denote the greatest of $k_1, ..., k_q, \mu_1, ..., \mu_q$ for program $\mathcal{P}$. Below, we use **k** as a shorthand for $k_1, ..., k_q$, and $\mu$ as a shorthand for $\mu_1, ..., \mu_q$. The equations in the algorithm come from the proof of Theorem 1, given in the sequel.

### Intuitionistic Modal Algorithm

**1.** Rename each modal atom $MA_j$ by a new atom not occurring in $\mathcal{P}$ of the form $A_j^{\Box}$ if $M = \Box$, or $A_j^{\Diamond}$ if $M = \Diamond$;

**2.** For each rule $r_l$ of the form $A_1, ..., A_k \Rightarrow A_0$ in $\mathcal{P}_i$ ($1 \leq i \leq n$) such that $\mathcal{R}(\omega_i, \omega_j)$, do: add a rule $A_1, ..., A_k \Rightarrow A_0$ to $\mathcal{P}_j$ ($1 \leq j \leq n$).

**3.** Calculate $A_{min} > (MAX_{\mathcal{P}}(\mathbf{k}, \mu, n) - 1) / (MAX_{\mathcal{P}}(\mathbf{k}, \mu, n) + 1)$;

**4.** Calculate $W \geq (2/\beta) \cdot (\ln(1 + A_{\min}) - \ln(1 - A_{\min})) / (MAX_{\mathcal{P}}(\mathbf{k}, \mu) \cdot (A_{\min} - 1) + A_{\min} + 1)$;

**5.** For each rule $r_l$ of the form $A_1, ..., A_k \Rightarrow A_0$ ($k \geq 0$) in $\mathcal{P}_i$ ($1 \leq i \leq n$), do:

(a) Add a neuron $N_l$ to the hidden layer of neural network $\mathcal{N}_i$ associated with $\mathcal{P}_i$; (b) Connect each neuron $A_i$ ($1 \leq i \leq k$) in the input layer of $\mathcal{N}_i$ to $N_l$ and set the connection weight to $W$; (c) Connect $N_l$ to neuron $A_0$ in the output layer of $\mathcal{N}_i$ and set the connection weight to $W$; (d) Set the threshold $\theta_l$ of $N_l$ to $\theta_l = ((1 + A_{\min}) \cdot (k_l - 1) / 2)W$; (e) Set the threshold $\theta_{A_0}$ of $A_0$ in the output layer of $\mathcal{N}_i$ to $\theta_{A_0} = ((1 + A_{\min}) \cdot (1 - \mu_l) / 2)W$. (f) For each atom of the form $A'$ in $r_l$, do:

(i) Add a hidden neuron $N_{A'}$ to $\mathcal{N}_i$; (ii) Set the step function $s(x)$ as the activation function of $N_{A'}$;[4] (iii) Set the threshold $\theta_{A'}$ of $N_{A'}$ such that $n - (1 + A_{min}) < \theta_{A'} < nA_{min}$; (iv) For each

network $\mathcal{N}_j$ corresponding to program $\mathcal{P}_j$ $(1 \leq j \leq n)$ in $\mathcal{P}$ such that $\mathcal{R}(\omega_i, \omega_j)$, do: Connect the output neuron $A$ of $\mathcal{N}_j$ to the hidden neuron $N_{A'}$ of $\mathcal{N}_i$ and set the connection weight to $-1$; and Connect the hidden neuron $N_{A'}$ of $\mathcal{N}_i$ to the output neuron $A'$ of $\mathcal{N}_i$ and set the connection weight to $W^I$ such that $W^I > h^{-1}(A_{min}) + \mu_{A'}.W + \theta_{A'}$.

**6.** For each output neuron $A_j^{\Diamond}$ in network $\mathcal{N}_i$, do:
(a) Add a hidden neuron $A_j^M$ and an output neuron $A_j$ to an arbitrary network $\mathcal{N}_z$ such that $\mathcal{R}(\omega_i, \omega_z)$; (b) Set the step function $s(x)$ as the activation function of $A_j^M$, and set the semi-linear function $h(x)$ as the activation function of $A_j$; (c) Connect $A_j^{\Diamond}$ in $\mathcal{N}_i$ to $A_j^M$ and set the connection weight to 1; (d) Set the threshold $\theta^M$ of $A_j^M$ such that $-1 < \theta^M < A_{min}$; (e) Set the threshold $\theta_{A_j}$ of $A_j$ in $\mathcal{N}_z$ such that $\theta_{A_j} = ((1 + A_{\min}) \cdot (1 - \mu_{A_j})/2)W$; (f) Connect $A_j^M$ to $A_j$ in $\mathcal{N}_z$ and set the connection weight to $W^M > h^{-1}(A_{\min}) + \mu_{A_j}W + \theta_{A_j}$.

**7.** For each output neuron $A_j^{\Box}$ in network $\mathcal{N}_i$, do:
(a) Add a hidden neuron $A_j^M$ to each $\mathcal{N}_u$ $(1 \leq u \leq n)$ such that $\mathcal{R}(\omega_i, \omega_u)$, and add an output neuron $A_j$ to $\mathcal{N}_u$ if $A_j \notin \mathcal{N}_u$; (b) Set the step function $s(x)$ as the activation function of $A_j^M$, and set the semi-linear function $h(x)$ as the activation function of $A_j$; (c) Connect $A_j^{\Box}$ in $\mathcal{N}_i$ to $A_j^M$ and set the connection weight to 1; (d) Set the threshold $\theta^M$ of $A_j^M$ such that $-1 < \theta^M < A_{min}$; (e) Set the threshold $\theta_{A_j}$ of $A_j$ in each $\mathcal{N}_u$ such that $\theta_{A_j} = ((1 + A_{\min}) \cdot (1 - \mu_{A_j})/2)W$; (f) Connect $A_j^M$ to $A_j$ in $\mathcal{N}_u$ and set the connection weight to $W^M > h^{-1}(A_{\min}) + \mu_{A_j}W + \theta_{A_j}$.

**8.** For each output neuron $A_j$ in network $\mathcal{N}_u$ such that $\mathcal{R}(\omega_i, \omega_u)$, do:
(a) Add a hidden neuron $A_j^{\vee}$ to $\mathcal{N}_i$; (b) Set the step function $s(x)$ as the activation function of $A_j^{\vee}$; (c) For each output neuron $A_j^{\Diamond}$ in $\mathcal{N}_i$, do:

(i) Connect $A_j$ in $\mathcal{N}_u$ to $A_j^{\vee}$ and set the connection weight to 1; (ii) Set the threshold $\theta^{\vee}$ of $A_j^{\vee}$ such that $-nA_{min} < \theta^{\vee} < A_{min} - (n-1)$; (iii) Connect $A_j^{\vee}$ to $A_j^{\Diamond}$ in $\mathcal{N}_i$ and set the connection weight to $W^M > h^{-1}(A_{\min}) + \mu_{A_j}W + \theta_{A_j}$.

**9.** For each output neuron $A_j$ in network $\mathcal{N}_u$ such that $\mathcal{R}(\omega_i, \omega_u)$, do:
(a) Add a hidden neuron $A_j^{\wedge}$ to $\mathcal{N}_i$; (b) Set the step function $s(x)$ as the activation function of $A_j^{\wedge}$; (c) For each output neuron $A_j^{\Box}$ in $\mathcal{N}_i$, do:

(i) Connect $A_j$ in $\mathcal{N}_u$ to $A_j^{\wedge}$ and set the connection weight to 1; (ii) Set the threshold $\theta^{\wedge}$ of $A_j^{\wedge}$ such that $n - (1 + A_{min}) < \theta^{\wedge} < nA_{min}$; (iii) Connect $A_j^{\wedge}$ to $A_j^{\Box}$ in $\mathcal{N}_i$ and set the connection weight to $W^M > h^{-1}(A_{\min}) + \mu_{A_j}W + \theta_{A_j}$.

Finally, we prove that $\mathcal{N}$ is equivalent to $\mathcal{P}$.

**Theorem 1** *(Correctness of Intuitionistic Modal Algorithm) For any intuitionistic modal program $\mathcal{P}$ there exists an ensemble of neural networks $\mathcal{N}$ such that $\mathcal{N}$ computes the intuitionistic modal semantics of $\mathcal{P}$.*
**Proof** *The algorithm to build each individual network in the ensemble is that of C-ILP, which we know is provably correct [3]. The algorithm to include modalities is that of CML, which is also provably correct [6]. We need to consider when modalities and intuitionistic negation are to be encoded together. Consider an output neuron $A_0$ with neurons $M$ (encoding modalities) and neurons $n$ (encoding negation) among its predecessors in a network's hidden layer. There are four cases to consider. (i) Both neurons $M$ and neurons $n$ are not activated: since the activation function of neurons $M$ and $n$ is the step function, their activation is zero, and thus this case reduces to C-ILP. (ii) Only neurons $M$ are activated: from the algorithm above, $A_0$ will also be activated (with minimum input potential $W^M + \varsigma$, where $\varsigma \in \mathbb{R}$). (iii) Only neurons $n$ are activated: as before, $A_0$ will also be activated (now with minimum input potential $W^I + \varsigma$). (iv) Both neurons $M$ and neurons $n$ are activated: the input potential of $A_0$ is at least $W^M + W^I + \varsigma$. Since $W^M > 0$ and $W^I > 0$, and since the activation function of $A_0$, $h(x)$, is monotonically increasing, $A_0$ will be activated whenever both $M$ and $n$ neurons are activated. This completes the proof.*

# 5    Concluding Remarks

In this paper, we have presented a new model of computation that integrates neural networks and constructive, intuitionistic modal reasoning. We have defined labelled intuitionistic modal programs, and have presented an algorithm to translate the intuitionistic theories into ensembles of C-ILP neural networks, and showed that the ensembles compute a semantics of the corresponding theories. As a result, each ensemble can be seen as a new massively parallel model for the computation of intuitionistic modal logic. In addition, since each network can be trained efficiently using, e.g., backpropagation, one can adapt the network ensemble by training possible world representations from examples. Work along these lines has been done in [4, 5], where learning experiments in possible worlds settings were investigated. As future work, we shall consider learning experiments based on the constructive model introduced in this paper. Extensions of this work also include the study of how to represent other non-classical logics such as branching time temporal logics, and conditional logics of normality, which are relevant for cognitive and neural computation.

**Acknowledgments**

Artur Garcez is partly supported by the Nuffield Foundation and The Royal Society. Luis Lamb is partly supported by the Brazilian Research Council CNPq and by the CAPES and FAPERGS foundations.

## Footnotes

[1]This is because if there were two white hats on their heads, one of them would have known (and have said), in the first round, that his hat was red, for he would have been seeing the other two with white hats.

[2]To complete the formalisation of the problem, the following rules should also hold at $t_2$ (and at $t_3$): $K_1 \neg p_2 \Rightarrow K_1 p_1$ and $K_1 \neg p_3 \Rightarrow K_1 p_1$. Analogous rules exist for agents 2 and 3.

[3]For example, if $A \wedge B \Rightarrow D$ and $C \Rightarrow D$ then a hidden neuron $h_1$ is used to connect $A$ and $B$ to $D$, and a hidden neuron $h_2$ is used to connect $C$ to $D$ such that if $h_1$ or $h_2$ is activated then $D$ is activated.

[4] Any hidden neuron created to encode negation (such as $h_4$ in Figure 1) shall have a non-linear activation function $s(x) = y$, where $y = 1$ if $x > 0$, and $y = 0$ otherwise. Such neurons encode (meta-level) knowledge about negation, while the other hidden neurons encode (object-level) knowledge about the problem domain. The former are not expected to be trained by examples and, as a result, the use of the step function will simplify the algorithm. The latter are to be trained, and therefore require a differentiable, semi-linear activation function.

# References

[1] A. Browne and R. Sun. Connectionist inference models. *Neural Networks*, 14(10):1331–1355, 2001.

[2] D. Van Dalen. Intuitionistic logic. In D. M. Gabbay and F. Guenthner, editors, *Handbook of Philosophical Logic*, volume 5. Kluwer, 2nd edition, 2002.

[3] A. S. d'Avila Garcez, K. Broda, and D. M. Gabbay. *Neural-Symbolic Learning Systems: Foundations and Applications*. Perspectives in Neural Computing. Springer-Verlag, 2002.

[4] A. S. d'Avila Garcez and L. C. Lamb. Reasoning about time and knowledge in neural-symbolic learning systems. In *Advances in Neural Information Processing Systems 16*, Proceedings of NIPS 2003, pages 921–928, Vancouver, Canada, 2004. MIT Press.

[5] A. S. d'Avila Garcez, L. C. Lamb, K. Broda, and D. M. Gabbay. Applying connectionist modal logics to distributed knowledge representation problems. *International Journal on Artificial Intelligence Tools*, 13(1):115–139, 2004.

[6] A. S. d'Avila Garcez, L. C. Lamb, and D. M. Gabbay. Connectionist modal logics. *Theoretical Computer Science*. Forthcoming.

[7] R. Fagin, J. Halpern, Y. Moses, and M. Vardi. *Reasoning about Knowledge*. MIT Press, 1995.

[8] D. M. Gabbay. *Labelled Deductive Systems*. Clarendom Press, Oxford, 1996.

[9] D. M. Gabbay, C. Hogger, and J. A. Robinson, editors. *Handbook of Logic in Artificial Intelligence and Logic Programming*, volume 1-5, Oxford, 1994-1999. Clarendom Press.

[10] M. Gelfond and V. Lifschitz. Classical negation in logic programs and disjunctive databases. *New Generation Computing*, 9:365–385, 1991.

[11] D. E. Rumelhart, G. E. Hinton, and R. J. Williams. Learning representations by back-propagating errors. *Nature*, 323:533–536, 1986.

[12] L. Shastri. Advances in SHRUTI: a neurally motivated model of relational knowledge representation and rapid inference using temporal synchrony. *Applied Intelligence*, 11:79–108, 1999.

[13] G. G. Towell and J. W. Shavlik. Knowledge-based artificial neural networks. *Artificial Intelligence*, 70(1):119–165, 1994.

[14] A. M. Turing. Computer machinery and intelligence. *Mind*, 59:433–460, 1950.

[15] L. G. Valiant. Robust logics. *Artificial Intelligence*, 117:231–253, 2000.

[16] V. Vapnik. *The nature of statistical learning theory*. Springer-Verlag, 1995.
